# Generalization Dynamics in
# LMS Trained Linear Networks

**Yves Chauvin***
Psychology Department
Stanford University
Stanford, CA 94305

## Abstract

For a simple linear case, a mathematical analysis of the training and generalization (validation) performance of networks trained by gradient descent on a Least Mean Square cost function is provided as a function of the learning parameters and of the statistics of the training data base. The analysis predicts that generalization error dynamics are very dependent on a *priori* initial weights. In particular, the generalization error might sometimes weave within a computable range during extended training. In some cases, the analysis provides bounds on the optimal number of training cycles for minimal validation error. For a speech labeling task, predicted weaving effects were qualitatively tested and observed by computer simulations in networks trained by the linear *and* non-linear back-propagation algorithm.

## 1 INTRODUCTION

Recent progress in network design demonstrates that non-linear feedforward neural networks can perform impressive pattern classification for a variety of real-world applications (e.g., Le Cun et al., 1990; Waibel et al., 1989). Various simulations and relationships between the neural network and machine learning theoretical literatures also suggest that too large a number of free parameters ("weight overfitting") could substantially reduce generalization performance. (e.g., Baum, 1989 1989).

A number of solutions have recently been proposed to decrease or eliminate the overfitting problem in specific situations. They range from *ad hoc* heuristics to theoretical considerations (e.g., Le Cun et al., 1990; Chauvin, 1990a; Weigend et al.,

In Press). For a phoneme labeling application, Chauvin showed that the overfitting phenomenon was actually observed only when networks were overtrained far beyond their "optimal" performance point (Chauvin, 1990b). Furthermore, generalization performance of networks seemed to be independent of the size of the network during early training but the rate of decrease in performance with overtraining was indeed related the number of weights.

The goal of this paper is to better understand training and generalization error dynamics in Least-Mean-Square trained *linear* networks. As we will see, gradient descent training on linear networks can actually generate surprisingly rich and insightful validation dynamics. Furthermore, in numerous applications, even non-linear networks tend to function in their linear range, as if the networks were making use of non-linearities only when necessary (Weigend et al., In Press; Chauvin, 1990a). In Section 2, I present a theoretical illustration yielding a better understanding of training and validation error dynamics. In Section 3, numerical solutions to obtained analytical results make interesting predictions for validation dynamics under overtraining. These predictions are tested for a phonemic labeling task. The obtained simulations suggest that the results of the analysis obtained with the simple theoretical framework of Section 2 might remain qualitatively valid for non-linear complex architectures.

# 2    THEORETICAL ILLUSTRATION

## 2.1    ASSUMPTIONS

Let us consider a linear network composed of $n$ input units and $n$ output units fully connected by a $n.n$ weight matrix $W$. Let us suppose the network is trained to reproduce a noiseless output "signal" from a noisy input "signal" (the network can be seen as a linear filter). We write $F$ as the "signal", $N$ the noise, $X$ the input, $Y$ the output, and $D$ the desired output. For the considered case, we have $X = F + N$, $Y = WX$ and $D = F$.

The statistical properties of the data base are the following. The signal is zero-mean with covariance matrix $C_F$. We write $\lambda_i$ and $e_i$ as the eigenvalues and eigenvectors of $C_F$ ($e_i$ are the so-called *principal components*; we will call $\lambda_i$ the "signal power spectrum"). The noise is assumed to be zero-mean, with covariance matrix $\bar{C}_N = \bar{\nu}.I$ where $I$ is the identity matrix. We assume the noise is uncorrelated with the signal: $\bar{C}_{FN} = 0$. We suppose two sets of patterns have been sampled for training and for validation. We write $C_F$, $C_N$ and $C_{FN}$ the resulting covariance matrices for the training set and $C'_F$, $C'_N$ and $C'_{FN}$ the corresponding matrices for the validation set. We assume $C_F \simeq C'_F \simeq \bar{C}_F$, $C_{FN} \simeq C'_{FN} \simeq \bar{C}_{FN} = 0$, $C_N = \nu.I$ and $C'_N = \nu'.I$ with $\nu' > \nu$. (Numerous of these assumptions are made for the sake of clarity of explanation: they can be relaxed without changing the resulting implications.)

The problem considered is much simpler than typical realistic applications. However, we will see below that (i) a formal analysis becomes complex very quickly (ii) the validation dynamics are rich, insightful and can be mapped to a number of results observed in simulations of realistic applications and (iii) an interesting number of predictions can be obtained.

## 2.2  LEARNING

The network is trained by gradient descent on the Least Mean Square (LMS) error: $\Delta W = -\eta \nabla_W E$ where $\eta$ is the usual learning rate and, in the case considered, $E = \sum_p^P (F_p - Y_p)^T (F_p - Y_p)$. We can write the gradient as a function of the various covariance matrices: $\nabla_W E = (I - W)C_F + (I - 2W)C_{FN} - WC_N$. From the general assumptions, we get:

$$\nabla_W E \simeq C_F - WC_F - WC_N \tag{1}$$

We assume now that the principal components $e_i$ are also eigenvectors of the weight matrix $W$ at iteration $k$ with corresponding eigenvalue $\alpha_{ik}$: $W_k.e_i = \alpha_{ik}e_i$. We can then compute the image of each eigenvector $e_i$ at iteration $k + 1$:

$$W_{k+1}.e_i = \eta\lambda_i.e_i + \alpha_{ik}[1 - \eta(\lambda_i + \nu)].e_i \tag{2}$$

Therefore, $e_i$ is also an eigenvector of $W_{k+1}$ and $\alpha_{i,k+1}$ satisfies the induction:

$$\alpha_{i,k+1} = \eta\lambda_i + \alpha_{ik}[1 - \eta(\lambda_i + \nu)] \tag{3}$$

Assuming $W_0 = 0$, we can compute the *alpha-dynamics* of the weight matrix $W$:

$$\alpha_{ik} = \frac{\lambda_i}{\lambda_i + \nu}[1 - (1 - \eta(\lambda_i + \nu))^k] \tag{4}$$

As $k$ goes to infinity, provided $\eta < 1/\lambda_M + \nu$, $\alpha_i$ approaches $\lambda_i/(\lambda_i + \nu_i)$, which corresponds to the optimal (Wiener) value of the linear filter implemented by the network. We will write the convergence rates $a_i = 1 - \eta\lambda_i - \eta\nu$. These rates depend on the signal "power spectrum", on the noise power and on the learning rate $\eta$.

If we now assume $W_0.e_i = \alpha_{i0}.e_i$ with $\alpha_{i0} \neq 0$ (this assumption can be made more general), we get:

$$\alpha_{ik} = \frac{\lambda_i}{\lambda_i + \nu}[1 - b_i a_i^k] \tag{5}$$

where $b_i = 1 - \alpha_{i0} - \alpha_{i0}\nu/\lambda_i$. Figure 1 represents possible alpha dynamics for arbitrary values of $\lambda_i$ with $\alpha_{i0} = \alpha_0 \neq 0$.

We can now compute the learning error dynamics by expanding the LMS error term $E$ at time $k$. Using the general assumptions on the covariance matrices, we find:

$$E_k = \sum_i^n E_{ik} = \sum_i^n \lambda_i(1 - \alpha_{ik})^2 + \nu\alpha_{ik}^2 \tag{6}$$

Therefore, training error is a sum of *error components*, each of them being a quadratic function of $\alpha_i$. Figure 2 represents a training error component $E_i$ as a function of $\alpha$. Knowing the *alpha-dynamics*, we can write these error components as a function of $k$:

$$E_{ik} = \frac{\lambda_i}{\lambda_i + \nu}(\nu + \lambda_i b^2 a^{2k}) \tag{7}$$

It is easy to see that $E$ is a monotonic decreasing function (generated by gradient descent) which converges to the bottom of the quadratic error surface, yielding the residual asymptotic error:

$$E_\infty = \sum_i^n \frac{\lambda_i\nu_i}{\lambda_i + \nu_i} \tag{8}$$

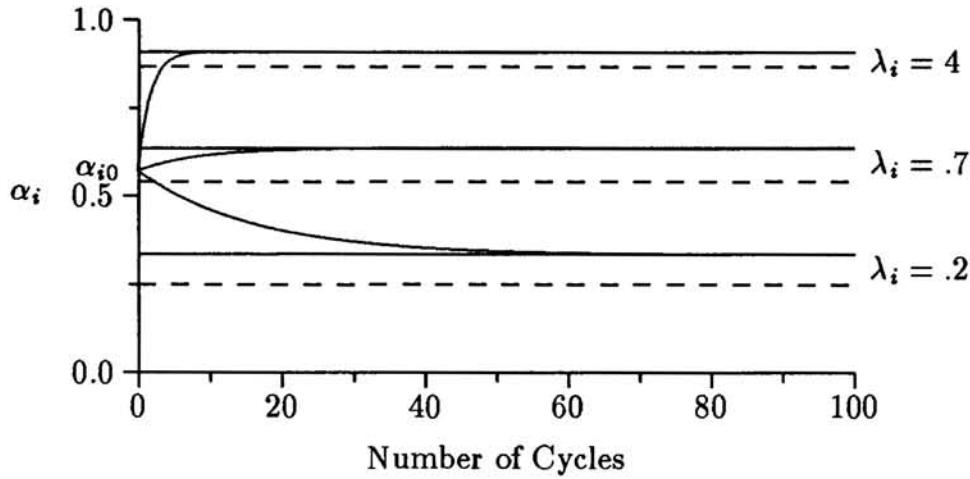

Figure 1: Alpha dynamics for different values of $\lambda_i$ with $\eta = .01$ and $\alpha_{i0} = \alpha_0 \neq 0$. The solid lines represent the optimal values of $\alpha_i$ for the training data set. The dashed lines represent corresponding optimal values for the validation data set.

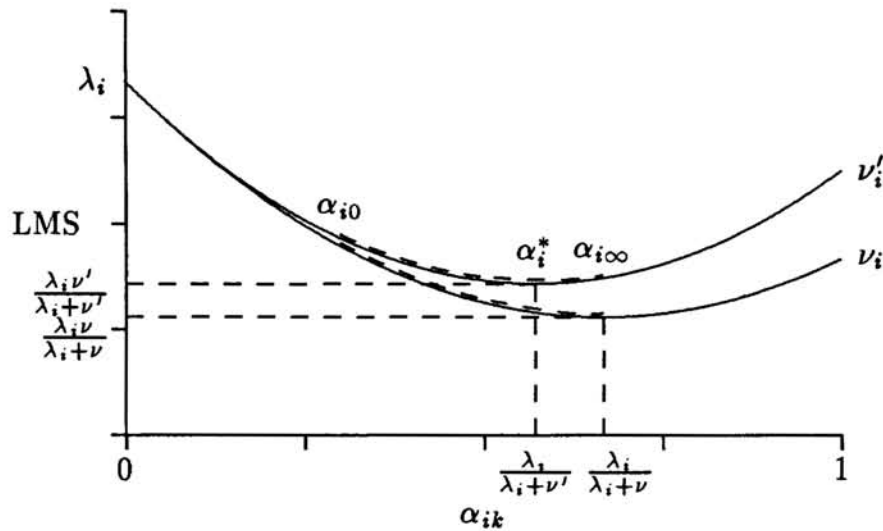

Figure 2: Training and validation error dynamics as a function of $\alpha_i$. The dashed curved lines represent the error dynamics for the initial conditions $\alpha_{i0}$. Each training error component follows the gradient of a quadratic learning curve (bottom). Note the overtraining phenomenon (top curve) between $\alpha_i^*$ (optimal for validation) and $\alpha_{i\infty}$ (optimal for training).

## 2.3  GENERALIZATION

Considering the general assumptions on the statistics of the data base, we can compute the validation error $E'$ (Note that "validation error" strictly applies to the validation data set. "Generalization error" can qualify the validation data set or the whole population, depending on context.):

$$E'_k = \sum_i^n E'_{ik} = \sum_i^n \lambda_i (1 - \alpha_{ik})^2 + \nu' \alpha_{ik}^2 \qquad (9)$$

where the alpha-dynamics are imposed by gradient descent learning on the training data set. Again, the validation error is a sum of error components $E'_i$, quadratic functions of $\alpha_i$. However, because the alpha-dynamics are adapted to the training sample, they might generate complex dynamics which will strongly depend on the inital values $\alpha_{i0}$ (Figure 1). Consequently, the resulting error components $E'_i$ are not monotonic decreasing functions anymore. As seen in Figure 2, each of the validation error components might (i) decrease (ii) decrease then increase (overtraining) or (iii) increase as a function of $\alpha_{i0}$. For each of these components, in the case of overtraining, it is possible to compute the value of $\alpha_{ik}$ at which training should be stopped to get minimal validation error:

$$k_i^* = \frac{Log \frac{\lambda_i}{\lambda_i + \nu'} + Log \frac{\nu' - \nu}{\lambda_i - \alpha_{i0}(\lambda_i + \nu')}}{Log(1 - \eta\lambda_i - \eta\nu)} \qquad (10)$$

However, the validation error dynamics become much more complex when we consider sums of these components. If we assume $\alpha_{i0} = 0$, the minimum (or minima) of $E'$ can be found to correspond to possible intersections of hyper-ellipsoids and power curves. In general, it is possible to show that there exists at least one such minimum. It is also possible to find simple bounds on the optimal training time for minimal validation error:

$$\frac{Log \frac{\nu' - \nu}{\lambda_m + \nu'}}{Log(1 - \eta\lambda_m - \eta\nu)} \leq k^* \leq \frac{Log \frac{\nu' - \nu}{\lambda_M + \nu'}}{Log(1 - \eta\lambda_M - \eta\nu)} \qquad (11)$$

These bounds are tight when the noise power is small compared to the signal "power spectrum". For $\alpha_{i0} \neq 0$, a formal analysis of the validation error dynamics becomes intractable. Because some error components might increase while others decrease, it is possible to imagine multiple minima and maxima for the total validation error (see simulations below). Considering each component's dynamics, it is nonetheless possible to compute bounds within which $E'$ might vary during training:

$$\sum_i^n \frac{\lambda_i \nu'}{\lambda_i + \nu'} \leq E'_k \leq \sum_i^n \frac{\lambda_i (\nu^2 + \nu' \lambda_i)}{(\lambda_i + \nu)^2} \qquad (12)$$

Because of the "exponential" nature of training (Figure 1), it is possible to imagine that this "weaving" effect might still be observed after a long training period, when the training error itself has become stable. Furthermore, whereas the training error will qualitatively show the same dynamics, validation error will very much depend on $\alpha_{i0}$: for sufficiently large initial weights, validation dynamics might be very dependent on particular simulation "runs".

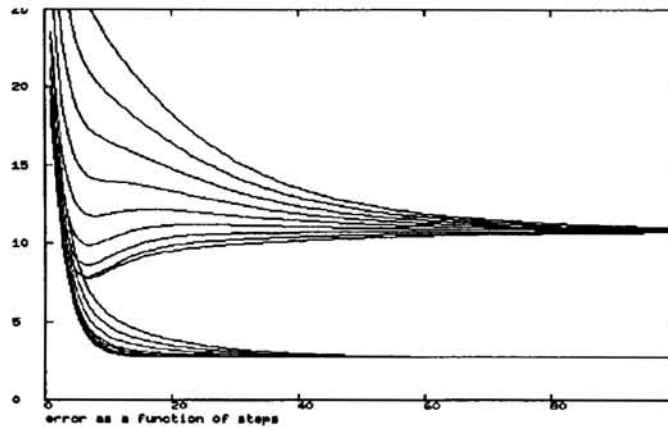

Figure 3: Training (bottom curves) and validation (top curves) error dynamics in a two-dimensional case for $\lambda_1 = 17, \lambda_2 = 1.7, \nu = 2, \nu' = 10, \alpha_{10} = 0$ as $\alpha_{20}$ varies from 0 to 1.6 (bottom-up) in .2 increments.

## 3   SIMULATIONS

### 3.1   CASE STUDY

Equations 7 and 9 were simulated for a two-dimensional case ($n = 2$) with $\lambda_1 = 17, \lambda_2 = 1.7, \nu = 2, \nu' = 10$ and $\alpha_{10} = 0$. The values of $\alpha_{20}$ determined the relative dominance of the two error components during training. Figure 3 represents training and validation dynamics as a function of $k$ for a range of values of $\alpha_{20}$. As shown analytically, training dynamics are basically unaffected by the initial conditions of the weight matrix $W_0$. However, a variety of validation dynamics can be observed as $\alpha_{20}$ varies from 0 to 1.6. For $1.6 \geq \alpha_{20} \geq 1.4$, the validation error is monotically decreasing and looks like a typical "gradient descent" training error. For $1.2 \geq \alpha_{20} \geq 1.0$, each error component in turn imposes a descent rate: the validation error looks like two "connected descents". For $.8 \geq \alpha_{20} \geq .6$, $E_2'$ is monotically decreasing with a slow convergence rate, forcing the validation error to decrease long after $E_1'$ has become stable. This creates a minimum, followed by a maximum, followed by a minimum for $E'$. Finally, for $.4 \geq \alpha_{20} \geq 0$, both error components have a single minimum during training and generate a single minimum for the total validation error $E'$.

### 3.2   PHONEMIC LABELING

One of the main predictions obtained from the analytical results and from the previous case study is that validation dynamics can demonstrate multiple local minima and maxima. To my knowledge, this phenomenon has not been described in the literature. However, the theory also predicts that the phenomenon will probably appear very late in training, well after the training error has become stable, which might explain the absence of such observations. The predictions were tested for a phonemic labeling task with spectrograms as input patterns and phonemes as output

patterns. Various architectures were tested (direct connections or back-propagation networks with linear or non-linear hidden layers). Due to the limited length of this article, the complete simulations will be reported elsewhere. In all cases, as predicted, multiple mimina/maxima were observed for the validation dynamics, provided the networks were trained way beyond usual training times. Furthermore, these generalization dynamics were very dependent on the initial weights (provided sufficient variance on the initial weight distribution).

## 4   DISCUSSION

It is sometimes assumed that optimal learning is obtained when validation error starts to increase during the course of training. Although for the theoretical study presented, the first minimum of $E'$ is probably always a global minimum, independently of $\alpha_{i0}$, simulations of the speech labeling task show it is not always the case with more complex architectures: late validation minima can sometimes (albeit rarely) be deeper than the first "local" minimum. These observations and a lack of theoretical understanding of statistical inference under limited data set raise the question of the significance of a validation data set. As a final comment, we are not really interested in minimal validation error $(E')$ but in minimal *generalization* error $(\bar{E}')$. Understanding the dynamics of the "population" error as a function of training and validation errors necessitates, at least, an evaluation of the sample statistics as a function of the number of training and validation patterns. This is beyond the scope of this paper.

**Acknowledgements**

Thanks to Pierre Baldi and Julie Holmes for their helpful comments.

**References**

Baum, E. B. & Haussler, D. (1989). What size net gives valid generalization? *Neural Computation, 1,* 151–160.

Chauvin, Y. (1990a). Dynamic behavior of constrained back-propagation networks. In D. S. Touretzky (Ed.), *Neural Information Processing Systems (Vol. 2)* (pp. 642–649). San Mateo, CA: Morgan Kaufman.

Chauvin, Y. (1990b). Generalization performance of overtrained back-propagation networks. In L. B. Almeida & C. J. Wellekens (Eds.), *Lecture Notes in Computer Science (Vol. 412)* (pp. 46–55). Berlin: Germany: Springer-Verlag.

Cun, Y. L., Boser, B., Denker, J. S., Henderson, D., Howard, R. E., Hubbard, W., & Jackel, L. D. (1990). Handwritten digit recognition with a back-propagation network. In D. S. Touretzky (Ed.), *Neural Information Processing Systems (Vol. 2)* (pp. 396–404). San Mateo, CA: Morgan Kaufman.

Waibel, A., Sawai, H., & Shikano, K. (1989). Modularity and scaling in large phonemic neural networks. *IEEE Transactions on Acoustics, Speech and Signal Processing, ASSP-37,* 1888–1898.

Weigend, A. S., Huberman, B. A., & Rumelhart, D. E. (In Press). Predicting the future: a connectionist approach. *International Journal of Neural Systems.*

## Footnotes

*Also with Thomson-CSF, Inc., 630 Hansen Way, Suite 250, Palo Alto, CA 94304.
